# Sparsity of SVMs that use the $\epsilon$-insensitive loss

**Ingo Steinwart**
Information Sciences Group CCS-3
Los Alamos National Laboratory
Los Alamos, NM 87545, USA
ingo@lanl.gov

**Andreas Christmann**
University of Bayreuth
Department of Mathematics
D-95440 Bayreuth
Andreas.Christmann@uni-bayreuth.de

## Abstract

In this paper lower and upper bounds for the number of support vectors are derived for support vector machines (SVMs) based on the $\epsilon$-insensitive loss function. It turns out that these bounds are asymptotically tight under mild assumptions on the data generating distribution. Finally, we briefly discuss a trade-off in $\epsilon$ between sparsity and accuracy if the SVM is used to estimate the conditional median.

## 1 Introduction

Given a reproducing kernel Hilbert space (RKHS) of a kernel $k : X \times X \to \mathbb{R}$ and training set $D := ((x_1, y_1), \ldots, (x_n, y_n)) \in (X \times \mathbb{R})^n$, the $\epsilon$-insensitive SVM proposed by Vapnik and his co-workers [10, 11] for regression tasks finds the unique minimizer $f_{D,\lambda} \in H$ of the regularized empirical risk

$$\lambda\|f\|_H^2 + \frac{1}{n}\sum_{i=1}^{n} L_\epsilon(y_i, f(x_i)),\tag{1}$$

where $L_\epsilon$ denotes the $\epsilon$-insensitive loss defined by $L_\epsilon(y, t) := \max\{0, |y - t| - \epsilon\}$ for all $y, t \in \mathbb{R}$ and some fixed $\epsilon \geq 0$. It is well known, see e.g. [2, Proposition 6.21], that the solution is of the form

$$f_{D,\lambda} = \sum_{i=1}^{n} \beta_i^* k(x_i, \cdot),\tag{2}$$

where the coefficients $\beta_i^*$ are a solution of the optimization problem

$$\text{maximize} \quad \sum_{i=1}^{n} y_i\beta_i - \epsilon\sum_{i=1}^{n}|\beta_i| - \frac{1}{2}\sum_{i,j=1}^{n}\beta_i\beta_j k(x_i, x_j)\tag{3}$$

$$\text{subject to} \quad -C \leq \beta_i \leq C \qquad \text{for all } i = 1, \ldots, n.\tag{4}$$

Here we set $C := 1/(2\lambda n)$. Note that the equality constraint $\sum_{i=1}^{n}\beta_i = 0$ needed in [2, Proposition 6.21] is superfluous since we do not include an offset term $b$ in the primal problem (1). In the following, we write $SV(f_{D,\lambda}) := \{i : \beta_i^* \neq 0\}$ for the set of indices that belong to the support vectors of $f_{D,\lambda}$. Furthermore, we write # for the counting measure, and hence $\#SV(f_{D,\lambda})$ denotes the number of support vectors of $f_{D,\lambda}$.

It is obvious from (2) that $\#SV(f_{D,\lambda})$ has a crucial influence on the time needed to compute $f_{D,\lambda}(x)$. Due to this fact, the $\epsilon$-insensitive loss was originally motivated by the goal to achieve *sparse* decision functions, i.e., decision functions $f_{D,\lambda}$ with $\#SV(f_{D,\lambda}) < n$. Although empirically it is well-known that the $\epsilon$-insensitive SVM achieves this sparsity, there is, so far, no theoretical explanation in the sense of [5]. The goal of this work is to provide such an explanation by establishing asymptotically tight lower and upper bounds for the number of support vectors. Based on these bounds we then investigate the trade-off between sparsity and estimation accuracy of the $\epsilon$-insensitive SVM.

## 2 Main results

Before we can formulate our main results we need to introduce some more notations. To this end, let P be a probability measure on $X \times \mathbb{R}$, where $X$ is some measurable space. Given a measurable $f : X \to \mathbb{R}$, we then define the $L_\epsilon$-risk of $f$ by $\mathcal{R}_{L_\epsilon,\mathrm{P}}(f) := \mathbb{E}_{(x,y)\sim\mathrm{P}} L_\epsilon(y, f(x))$. Moreover, recall that P can be split into the marginal distribution $\mathrm{P}_X$ on $X$ and the regular conditional probability $\mathrm{P}(\,\cdot\,|x)$. Given a RKHS $H$ of a bounded kernel $k$, [1] then showed that

$$f_{\mathrm{P},\lambda} := \arg\inf_{f\in H} \lambda\|f\|_H^2 + \mathcal{R}_{L_\epsilon,\mathrm{P}}(f)$$

exists and is uniquely determined whenever $\mathcal{R}_{L_\epsilon,\mathrm{P}}(0) < \infty$. Let us write $\delta_{(x,y)}$ for the Dirac measure at some $(x,y) \in X \times \mathbb{R}$. By considering the empirical measure $\mathrm{D} := \frac{1}{n} \sum_{i=1}^n \delta_{(x_i,y_i)}$ of a training set $D := ((x_1,y_1),\ldots,(x_n,y_n)) \in (X \times \mathbb{R})^n$, we then see that the corresponding $f_{\mathrm{D},\lambda}$ is the solution of (1). Finally, we need to introduce the sets

$$
\begin{aligned}
A_{\mathrm{low}}^\delta(f) &:= \{(x,y) \in X \times \mathbb{R} : |f(x) - y| > \epsilon + \delta\} \\
A_{\mathrm{up}}^\delta(f) &:= \{(x,y) \in X \times \mathbb{R} : |f(x) - y| \geq \epsilon - \delta\},
\end{aligned}
$$

where $f : X \to \mathbb{R}$ is an arbitrary function and $\delta \in \mathbb{R}$. Moreover, we use the short forms $A_{\mathrm{low}}(f) := A_{\mathrm{low}}^0(f)$ and $A_{\mathrm{up}}(f) := A_{\mathrm{up}}^0(f)$. Now we can formulate our first main result.

**Theorem 2.1** *Let* P *be a probability measure on* $X \times \mathbb{R}$ *and* $H$ *be a separable RKHS with bounded measurable kernel satisfying* $\|k\|_\infty \leq 1$. *Then, for all* $n \geq 1$, $\rho > 0$, $\delta > 0$, *and* $\lambda > 0$ *satisfying* $\delta\lambda \leq 4$, *we have*

$$\mathrm{P}^n\Big( D \in (X \times \mathbb{R})^n : \frac{\#SV(f_{\mathrm{D},\lambda})}{n} > \mathrm{P}\big(A_{\mathrm{low}}^\delta(f_{\mathrm{P},\lambda})\big) - \rho \Big) \geq 1 - 3e^{-\frac{\delta^2\lambda^2 n}{16}} - e^{-2\rho^2 n}$$

*and*

$$\mathrm{P}^n\Big( D \in (X \times \mathbb{R})^n : \frac{\#SV(f_{\mathrm{D},\lambda})}{n} < \mathrm{P}\big(A_{\mathrm{up}}^\delta(f_{\mathrm{P},\lambda})\big) + \rho \Big) \geq 1 - 3e^{-\frac{\delta^2\lambda^2 n}{16}} - e^{-2\rho^2 n}\,.$$

Before we present our second main result, we briefly illustrate Theorem 2.1 for the case where we fix the regularization parameter $\lambda$ and let $n \to \infty$.

**Corollary 2.2** *Let* P *be a probability measure on* $X \times \mathbb{R}$ *and* $H$ *be a separable RKHS with bounded measurable kernel satisfying* $\|k\|_\infty \leq 1$. *Then, for all* $\rho > 0$ *and* $\lambda > 0$, *we have*

$$\lim_{n\to\infty} \mathrm{P}^n\Big( D \in (X \times \mathbb{R})^n : \mathrm{P}\big(A_{\mathrm{low}}(f_{\mathrm{P},\lambda})\big) - \rho \leq \frac{\#SV(f_{\mathrm{D},\lambda})}{n} \leq \mathrm{P}\big(A_{\mathrm{up}}(f_{\mathrm{P},\lambda})\big) + \rho \Big) = 1\,.$$

Note that the above corollary exactly describes the asymptotic behavior of the fraction of support vectors modulo the probability of the set

$$A_{\mathrm{up}}(f_{\mathrm{P},\lambda}) \backslash A_{\mathrm{low}}(f_{\mathrm{P},\lambda}) = \big\{(x, f_{\mathrm{P},\lambda}(x) - \epsilon) : x \in X\big\} \cup \big\{(x, f_{\mathrm{P},\lambda}(x) + \epsilon) : x \in X\big\}\,.$$

In particular, if the conditional distributions $\mathrm{P}(\,\cdot\,|x)$, $x \in X$, have no discrete components, then the above corollary gives an exact description.

Of course, in almost no situation it is realistic to assume that $\lambda$ stays fixed if the sample size $n$ grows. Instead, it is well-known, see [1], that the regularization parameter should vanish in order to achieve consistency. To investigate this case, we need to introduce some additional notations from [6] that are related to the $L_\epsilon$-risk. Let us begin by denoting the Bayes $L_\epsilon$-risk by $\mathcal{R}_{L_\epsilon,\mathrm{P}}^* := \inf \mathcal{R}_{L_\epsilon,\mathrm{P}}(f)$, where P is a distribution and the infimum is taken over all measurable functions $f : X \to \mathbb{R}$. In addition, given a distribution Q on $\mathbb{R}$, [6] and [7, Chapter 3] defined the *inner* $L_\epsilon$-*risks* by

$$\mathcal{C}_{L_\epsilon,\mathrm{Q}}(t) := \int_{\mathbb{R}} L_\epsilon(y, t)\, d\mathrm{Q}(y), \qquad t \in \mathbb{R},$$

and the *minimal inner* $L_\epsilon$-*risks* were denoted by $\mathcal{C}_{L_\epsilon,\mathrm{Q}}^* := \inf_{t\in\mathbb{R}} \mathcal{C}_{L_\epsilon,\mathrm{Q}}(t)$. Obviously, we have

$$\mathcal{R}_{L_\epsilon,\mathrm{P}}(f) = \int_X \mathcal{C}_{L_\epsilon,\mathrm{P}(\,\cdot\,|x)}\big(f(x)\big)\, d\mathrm{P}_X(x)\,, \tag{5}$$

and [6, Lemma 2.5], see also [7, Lemma 3.4], further established the intuitive formula $\mathcal{R}^*_{L_\epsilon,\mathrm{P}} = \int_X \mathcal{C}^*_{L_\epsilon,\mathrm{P}(\,\cdot\,|x)}\, d\mathrm{P}_X(x)$. Moreover, we need the sets of *conditional minimizers*

$$\mathcal{M}^*(x) := \left\{ t \in \mathbb{R} : \mathcal{C}_{L_\epsilon,\mathrm{P}(\,\cdot\,|x)}(t) = \mathcal{C}^*_{L_\epsilon,\mathrm{P}(\,\cdot\,|x)} \right\}.$$

The following lemma collects some useful properties of these sets.

**Lemma 2.3** *Let* $\mathrm{P}$ *be a probability measure on* $X \times \mathbb{R}$ *with* $\mathcal{R}^*_{L_\epsilon,\mathrm{P}} < \infty$. *Then* $\mathcal{M}^*(x)$ *is a non-empty and compact interval for* $\mathrm{P}_X$*-almost all* $x \in X$.

Given a function $f : X \to \mathbb{R}$, Lemma 2.3 shows that for $\mathrm{P}_X$-almost all $x \in X$ there exists a unique $t^*(x) \in \mathcal{M}^*(x)$ such that

$$\left| t^*(x) - f(x) \right| \leq \left| t - f(x) \right| \qquad \text{for all } t \in \mathcal{M}^*(x). \tag{6}$$

In other words, $t^*(x)$ is the element in $\mathcal{M}^*(x)$ that has the smallest distance to $f(x)$. In the following, we sometimes write $t^*_\lambda(x) := t^*(x)$ if $f = f_{\mathrm{P},\lambda}$ and we wish to emphasize the dependence of $t^*(x)$ on $\lambda$. With the help of these elements, we finally introduce the sets

$$\begin{aligned} M^\delta_{\mathrm{low}}(f) &:= \left\{ (x,y) \in X \times \mathbb{R} : |t^*(x) - y| > \epsilon + \delta \right\} \\ M^\delta_{\mathrm{up}}(f) &:= \left\{ (x,y) \in X \times \mathbb{R} : |t^*(x) - y| \geq \epsilon - \delta \right\}, \end{aligned}$$

where $\delta \in \mathbb{R}$. Moreover, we again use the short forms $M_{\mathrm{low}}(f) := M^0_{\mathrm{low}}(f)$ and $M_{\mathrm{up}}(f) := M^0_{\mathrm{up}}(f)$. Now we can formulate our second main result.

**Theorem 2.4** *Let* $\mathrm{P}$ *be a probability measure on* $X \times \mathbb{R}$ *and* $H$ *be a separable RKHS with bounded measurable kernel satisfying* $\|k\|_\infty \leq 1$. *Assume that* $\mathcal{R}_{L_\epsilon,\mathrm{P}}(0) < \infty$ *and that* $H$ *is dense in* $L_1(\mathrm{P}_X)$. *Then, for all* $\rho > 0$, *there exist a* $\delta_\rho > 0$ *and a* $\lambda_\rho > 0$ *such that for all* $\lambda \in (0, \lambda_\rho]$ *and all* $n \geq 1$ *we have*

$$\mathrm{P}^n\left( D \in (X \times \mathbb{R})^n : \mathrm{P}\big(M_{\mathrm{low}}(f_{\mathrm{P},\lambda})\big) - \rho \leq \frac{\#SV(f_{\mathrm{D},\lambda})}{n} \leq \mathrm{P}\big(M_{\mathrm{up}}(f_{\mathrm{P},\lambda})\big) + \rho \right) \geq 1 - 8e^{-\delta_\rho^2 \lambda^2 n}.$$

If we choose a sequence of regularization parameters $\lambda_n$ such that $\lambda_n \to 0$ and $\lambda_n^2 n \to \infty$, then the resulting SVM is $L_\epsilon$-risk consistent under the assumptions of Theorem 2.4, see [1]. For this case, the following obvious corollary of Theorem 2.4 establishes lower and upper bounds on the number of support vectors.

**Corollary 2.5** *Let* $\mathrm{P}$ *be a probability measure on* $X \times \mathbb{R}$ *and* $H$ *be a separable RKHS with bounded measurable kernel satisfying* $\|k\|_\infty \leq 1$. *Assume that* $\mathcal{R}_{L_\epsilon,\mathrm{P}}(0) < \infty$ *and that* $H$ *is dense in* $L_1(\mathrm{P}_X)$. *Furthermore, let* $(\lambda_n) \in (0, \infty)$ *be a sequence with* $\lambda_n \to 0$ *and* $\lambda_n^2 n \to \infty$. *Then, for all* $\rho > 0$, *the probability* $\mathrm{P}^n$ *of* $D \in (X \times \mathbb{R})^n$ *satisfying*

$$\liminf_{m \to \infty} \mathrm{P}\big(M_{\mathrm{low}}(f_{\mathrm{P},\lambda_m})\big) - \rho \leq \frac{\#SV(f_{\mathrm{D},\lambda_n})}{n} \leq \limsup_{m \to \infty} \mathrm{P}\big(M_{\mathrm{up}}(f_{\mathrm{P},\lambda_m})\big) + \rho$$

*converges to* 1 *for* $n \to \infty$.

In general, the probabilities of the sets $M_{\mathrm{low}}(f_{\mathrm{P},\lambda})$ and $M_{\mathrm{up}}(f_{\mathrm{P},\lambda})$ are hard to control since, e.g., for fixed $x \in X$ and $\lambda \to 0$ it seems difficult to show that $f_{\mathrm{P},\lambda}(x)$ is *not* "flipping" from the left hand side of $\mathcal{M}^*(x)$ to the right hand side. Indeed, for general $\mathcal{M}^*(x)$, such flipping would give different values $t^*_\lambda(x) \in \mathcal{M}^*(x)$ for $\lambda \to 0$, and hence would result in significantly different sets $M_{\mathrm{low}}(f_{\mathrm{P},\lambda})$ and $M_{\mathrm{up}}(f_{\mathrm{P},\lambda})$. As a consequence, it seems hard to show that, for probability measures $\mathrm{P}$ whose conditional distributions $\mathrm{P}(\,\cdot\,|x)$, $x \in X$, have no discrete components, we always have

$$\liminf_{\lambda \to 0} \mathrm{P}\big(M_{\mathrm{low}}(f_{\mathrm{P},\lambda})\big) = \limsup_{\lambda \to 0} \mathrm{P}\big(M_{\mathrm{up}}(f_{\mathrm{P},\lambda})\big). \tag{7}$$

However, there are situations in which this equality can easily be established. For example, assume that the sets $\mathcal{M}^*(x)$ are $\mathrm{P}_X$-almost surely singletons. In this case, $t^*_\lambda(x)$ is in fact *independent* of $\lambda$, and hence so are $M_{\mathrm{low}}(f_{\mathrm{P},\lambda})$ and $M_{\mathrm{up}}(f_{\mathrm{P},\lambda})$. Namely, in this case these sets contain the pairs $(x,y)$

for which $y$ is *not* contained in the closed or open $\epsilon$-tube around $\mathcal{M}^*(x)$, respectively. Consequently, (7) holds provided that the conditional distributions $\mathrm{P}(\,\cdot\,|x)$, $x \in X$, have no discrete components, and hence Corollary 2.5 gives a tight bound on the number of support vectors. Moreover, if in this case we additionally assume $\epsilon = 0$, i.e., we consider the absolute loss, then we easily find $\mathrm{P}(M_{\mathrm{low}}(f_{\mathrm{P},\lambda})) = \mathrm{P}(M_{\mathrm{up}}(f_{\mathrm{P},\lambda})) = 1$, and hence Corollary 2.5 shows that the corresponding SVM does *not* tend to produce sparse decision functions. Finally, recall that for this specific loss function, $\mathcal{M}^*(x)$ equals the median of $\mathrm{P}(\,\cdot\,|x)$, and hence $\mathcal{M}^*(x)$ is a singleton whenever the median of $\mathrm{P}(\,\cdot\,|x)$ is unique.

Let us now illustrate Corollary 2.5 for $\epsilon > 0$. To this end, we assume in the following that the conditional distributions $\mathrm{P}(\,\cdot\,|x)$ are *symmetric*, i.e., for $\mathrm{P}_X$-almost all $x \in X$ there exists a *conditional center* $c(x) \in \mathbb{R}$ such that $\mathrm{P}(c(x) + A|x) = \mathrm{P}(c(x) - A|x)$ for all measurable $A \subset \mathbb{R}$. Note that by considering $A := [0, \infty)$ it is easy to see that $c(x)$ is a median of $\mathrm{P}(\,\cdot\,|x)$. Furthermore, the assumption $\mathcal{R}_{L_\epsilon,\mathrm{P}}(0) < \infty$ imposed in the results above ensures that the conditional mean $f_\mathrm{P}^*(x) := \mathbb{E}(Y|x)$ of $\mathrm{P}(\,\cdot\,|x)$ exists $\mathrm{P}_X$-almost surely, and from this it is easy to conclude that $c(x) = f_\mathrm{P}^*(x)$ for $\mathrm{P}_X$-almost all $x \in X$. Moreover, from [8, Proposition 3.2 and Lemma 3.3] we immediately obtain the following lemma.

**Lemma 2.6** *Let* $\mathrm{P}$ *be a probability measure on* $X \times \mathbb{R}$ *such that* $\mathcal{R}_{L_\epsilon,\mathrm{P}}(0) < \infty$. *Assume that the conditional distributions* $\mathrm{P}(\,\cdot\,|x)$, $x \in X$, *are symmetric and that for* $\mathrm{P}_X$-*almost all* $x \in X$ *there exists a* $\delta(x) > 0$ *such that for all* $\delta \in (0, \delta(x)]$ *we have*

$$\mathrm{P}\big(f_\mathrm{P}^*(x) + [-\delta, \delta]\big|x\big) \;>\; 0\,, \tag{8}$$

$$\mathrm{P}\big(f_\mathrm{P}^*(x) + [\epsilon - \delta, \epsilon + \delta]\big|x\big) \;>\; 0\,. \tag{9}$$

*Then, for* $\mathrm{P}_X$-*almost all* $x \in X$, *we have* $\mathcal{M}^*(x) = \{f_\mathrm{P}^*(x)\}$ *and* $f_\mathrm{P}^*(x)$ *equals* $\mathrm{P}_X$-*almost surely the unique median of* $\mathrm{P}(\,\cdot\,|x)$.

Obviously, condition (8) means that the conditional distributions have some mass around their median $f_\mathrm{P}^*$, whereas (9) means that the conditional distributions have some mass around $f_\mathrm{P}^* \pm \epsilon$. Moreover, [8] showed that under the assumptions of Lemma 2.6, the corresponding $\epsilon$-insensitive SVM can be used to estimate the conditional median. Let us now illustrate how the value of $\epsilon$ influences both the accuracy of this estimate and the sparsity. To this end, let us assume for the sake of simplicity that the conditional distributions $\mathrm{P}(\,\cdot\,|x)$ have *continuous* Lebesgue densities $p(\,\cdot\,|x) : \mathbb{R} \to [0, \infty)$. By the symmetry of the conditional distributions it is then easy to see that these densities are symmetric around $f_\mathrm{P}^*(x)$. Now, it follows from the continuity of the densities, that (8) is satisfied if $p(f_\mathrm{P}^*(x)|x) > 0$, whereas (9) is satisfied if $p(f_\mathrm{P}^*(x) + \epsilon|x) > 0$. Let us first consider the case where the conditional distributions are equal modulo translations. In other words, we assume that there exists a continuous Lebesgue density $q : \mathbb{R} \to [0, \infty)$ which is symmetric around 0 such that for $\mathrm{P}_X$-almost all $x \in X$ we have

$$q(y) = p(f_\mathrm{P}^*(x) + y|x)\,, \qquad y \in \mathbb{R}.$$

Note that this assumption is essentially identical to a classical "signal plus noise" assumption. In the following we further assume that $q$ is unimodal, i.e., $q$ has its only local and global maximum at 0. From this we easily see that (8) is satisfied, and (9) is satisfied if $q(\epsilon) > 0$. By Lemma 2.6 and the discussion around (7) we then conclude that under the assumptions of Corollary 2.5 the fraction of support vectors asymptotically approaches $2Q([\epsilon, \infty))$, where $Q$ is the probability measure defined by $q$. This confirms the intuition that *larger values of* $\epsilon$ *lead to sparser decision functions*. In particular, if $Q([\epsilon, \infty)) = 0$, the corresponding SVM produces *super sparse* decision functions, i.e., decision functions whose number of support vectors does *not* grow linearly in the sample size. However, not surprisingly, there is a price to be paid for this sparsity. Indeed, [8, Lemma 3.3] indicates that the size of $q(\epsilon)$ has a direct influence on the ability of $f_{\mathrm{D},\lambda}$ to estimate the conditional median $f_\mathrm{P}^*$. Let us describe this in a little more detail. To this end, we first find by [8, Lemma 3.3] and the convexity of $t \mapsto \mathcal{C}_{L_\epsilon,Q}(t)$ that

$$\mathcal{C}_{L_\epsilon,Q}(t) - \mathcal{C}_{L_\epsilon,Q}^* \geq q(\epsilon) \cdot \begin{cases} t^2/2 & \text{if } t \in [0, \epsilon] \\ t\epsilon - \epsilon^2/2 & \text{if } t \geq \epsilon. \end{cases}$$

By a literal repetition of the proof of [8, Theorem 2.5] we then find the self-calibration inequality

$$\|f - f_\mathrm{P}^*\|_{L_1(\mathrm{P}_X)} \leq \sqrt{2/q(\epsilon)}\sqrt{\mathcal{R}_{L_\epsilon,\mathrm{P}}(f) - \mathcal{R}_{L_\epsilon,\mathrm{P}}^*}\,, \tag{10}$$

which holds for all $f : X \to \mathbb{R}$ with $\mathcal{R}_{L_\epsilon,\mathrm{P}}(f) - \mathcal{R}^*_{L_\epsilon,\mathrm{P}} \leq \epsilon^2/2$. Now, if we are in the situation of Corollary 2.5, then we know that $\mathcal{R}_{L_\epsilon,\mathrm{P}}(f_{\mathrm{D},\lambda_n}) \to \mathcal{R}^*_{L_\epsilon,\mathrm{P}}$ in probability for $n \to \infty$, and thus (10) shows that $f_{\mathrm{D},\lambda_n}$ approximates the conditional median $f^*_\mathrm{P}$ with respect to the $L_1(\mathrm{P}_X)$-norm. However, the guarantee for this approximation becomes worse the smaller $q(\epsilon)$ becomes, i.e., the larger $\epsilon$ is. In other words, *the sparsity of the decision functions may be paid by less accurate estimates of the conditional median.* On the other hand, our results also show that *moderate values for $\epsilon$ can lead to both reasonable estimates of the conditional median and relatively sparse decision functions.* In this regard we further note that one can also use [8, Lemma 3.3] to establish self-calibration inequalities that measure the distance of $f$ to $f^*_\mathrm{P}$ only up to $\epsilon$. In this case, however, it is obvious that such self-calibration inequalities are worse the larger $\epsilon$ is, and hence the informal conclusions above remain unchanged.

Finally, we like to mention that, if the conditional distributions are not equal modulo translations, then the situation may become more involved. In particular, if we are in a situation with $p(f^*_\mathrm{P}(x)|x) > 0$ and $p(f^*_\mathrm{P}(x) + \epsilon|x) > 0$ but $\inf_x p(f^*_\mathrm{P}(x)|x) = \inf_x p(f^*_\mathrm{P}(x) + \epsilon|x) = 0$, self-calibration inequalities of the form (10) are in general impossible, and weaker self-calibration inequalities require additional assumptions on P. We refer to [8] where the case $\epsilon = 0$ is considered.

## 3 Proofs

Setting $C := \frac{1}{2\lambda n}$ and introducing slack variables, we can restate the optimization problem (1) as

$$\text{minimize} \quad \frac{1}{2}\|f\|^2_H + C\sum_{i=1}^{n}(\xi_i + \tilde{\xi}_i) \tag{11}$$

$$\text{subject to} \quad f(x_i) - y_i \leq \epsilon + \xi_i,$$
$$y_i - f(x_i) \leq \epsilon + \tilde{\xi}_i,$$
$$\xi_i, \tilde{\xi}_i \geq 0 \qquad \text{for all } i = 1, \dots, n.$$

In the following we denote the (unique) solution of (11) by $(f^*, \xi^*, \tilde{\xi}^*)$, where we note that we have $f^* = f_{\mathrm{D},\lambda}$. It is well-known, see e.g. [2, p. 117], that the dual optimization problem of (11) is

$$\text{maximize} \quad \sum_{i=1}^{n} y_i(\tilde{\alpha}_i - \alpha_i) - \epsilon \sum_{i=1}^{n}(\tilde{\alpha}_i + \alpha_i) - \frac{1}{2}\sum_{i,j=1}^{n}(\tilde{\alpha}_i - \alpha_i)(\tilde{\alpha}_j - \alpha_j)k(x_i, x_j) \tag{12}$$

$$\text{subject to} \quad 0 \leq \alpha_i, \tilde{\alpha}_i \leq C \qquad \text{for all } i = 1, \dots, n,$$

where $k$ is the kernel of the RKHS $H$. Furthermore, if $(\alpha^*_1, \tilde{\alpha}^*_1, \dots, \alpha^*_n, \tilde{\alpha}^*_n)$ denotes a solution of (12), then we can recover the primal solution $(f^*, \xi^*, \tilde{\xi}^*)$ by

$$f^* = \sum_{i=1}^{n}(\tilde{\alpha}^*_i - \alpha^*_i)k(x_i, \cdot), \tag{13}$$

$$\xi^*_i = \max\{0, f^*(x_i) - y_i - \epsilon\}, \tag{14}$$

$$\tilde{\xi}^*_i = \max\{0, y_i - f^*(x_i) - \epsilon\}, \tag{15}$$

for all $i = 1, \dots, n$. Moreover, the Karush-Kuhn-Tucker conditions of (12) are

$$\alpha^*_i(f^*(x_i) - y_i - \epsilon - \xi^*_i) = 0, \tag{16}$$

$$\tilde{\alpha}^*_i(y_i - f^*(x_i) - \epsilon - \tilde{\xi}^*_i) = 0, \tag{17}$$

$$(\alpha^*_i - C)\xi^*_i = 0, \tag{18}$$

$$(\tilde{\alpha}^*_i - C)\tilde{\xi}^*_i = 0, \tag{19}$$

$$\xi^*_i\tilde{\xi}^*_i = 0, \tag{20}$$

$$\alpha^*_i\tilde{\alpha}^*_i = 0, \tag{21}$$

where $i = 1, \dots, n$. Finally, note that by setting $\beta_i := \tilde{\alpha}_i - \alpha_i$ the problem (12) can be simplified to (3), and consequently, a solution $\beta^*$ of (3) is of the form $\beta^* = \tilde{\alpha}^* - \alpha^*$. The following simple lemma provides lower and upper bounds for the set of support vectors.

**Lemma 3.1** *Using the above notations we have*
$$\left\{ i : |f_{D,\lambda}(x_i) - y_i| > \epsilon \right\} \subset \left\{ i : \beta_i^* \neq 0 \right\} \subset \left\{ i : |f_{D,\lambda}(x_i) - y_i| \geq \epsilon \right\}.$$

***Proof:*** Let us first prove the inclusion on the left hand side. To this end, we begin by fixing an index $i$ with $f_{D,\lambda}(x_i) - y_i > \epsilon$. By $f_{D,\lambda} = f^*$ and (14), we then find $\xi_i^* > 0$, and hence (18) implies $\alpha_i^* = C$. From (21) we conclude $\tilde{\alpha}_i^* = 0$ and hence we have $\beta_i^* = \tilde{\alpha}_i^* - \alpha_i^* = -C \neq 0$. The case $y_i - f_{D,\lambda}(x_i) > \epsilon$ can be shown analogously, and hence we obtain the first inclusion. In order to show the second inclusion we fix an index $i$ with $\beta_i^* \neq 0$. By $\beta_i^* = \tilde{\alpha}_i^* - \alpha_i^*$ and (21) we then have *either* $\alpha_i^* \neq 0$ *or* $\tilde{\alpha}_i^* \neq 0$. Let us first consider the case $\alpha_i^* \neq 0$ and $\tilde{\alpha}_i^* = 0$. The KKT condition (16) together with $f_{D,\lambda} = f^*$ implies $f_{D,\lambda}(x_i) - y_i - \epsilon = \xi_i^*$ and since $\xi_i^* \geq 0$ we get $f_{D,\lambda}(x_i) - y_i \geq \epsilon$. The second case $\tilde{\alpha}_i^* = 0$ can be shown analogously. ∎

We further need the following Hilbert space version of Hoeffding's inequality from [12, Chapter 3], see also [7, Chapter 6.2] for a slightly sharper inequality.

**Theorem 3.2** *Let $(\Omega, \mathcal{A}, P)$ be a probability space and $H$ be a separable Hilbert space. Moreover, let $\eta_1, \ldots, \eta_n : \Omega \to H$ be independent random variables satisfying $\mathbb{E}_P \eta_i = 0$ and $\|\eta_i\|_\infty \leq 1$ for all $i = 1, \ldots, n$. Then, for all $\tau \geq 1$ and all $n \geq \tau$, we have*
$$P\left( \left\| \frac{1}{n} \sum_{i=1}^n \eta_i \right\|_H < 4\sqrt{\frac{\tau}{n}} \right) \geq 1 - 3e^{-\tau}.$$

Finally, we need the following theorem, see [7, Corollary 5.10], which was essentially shown by [13, 5, 3].

**Theorem 3.3** *Let $P$ be a probability measure on $X \times \mathbb{R}$ and $H$ be a separable RKHS with bounded measurable kernel satisfying $\|k\|_\infty \leq 1$. We write $\Phi : X \to H$ for the canonical feature map of $H$, i.e., $\Phi(x) := k(\,\cdot\,, x)$, $x \in X$. Then for all $\lambda > 0$ there exists a function $h : X \times \mathbb{R} \to [-1, 1]$ such that for all $n \geq 1$ and all $D \in (X \times \mathbb{R})^n$ we have*
$$\|f_{D,\lambda} - f_{P,\lambda}\|_H \leq \lambda^{-1} \|\mathbb{E}_D h\Phi - \mathbb{E}_P h\Phi\|_H,$$
*where $\mathbb{E}_D$ denotes the empirical average with respect to $D$.*

***Proof of of Theorem 2.1:*** In order to show the first estimate we fix a $\delta > 0$ and a $\lambda > 0$ such that $\delta\lambda \leq 4$. Let $\tau := \lambda^2 \delta^2 n / 16$ which implies $n \geq \tau$. Combining Theorems 3.2 and 3.3 we then obtain
$$
\begin{aligned}
1 - 3e^{-\tau} &\leq P^n\left( D \in (X \times \mathbb{R})^n : \|\mathbb{E}_D h\Phi - \mathbb{E}_P h\Phi\|_H \leq 4\sqrt{\tau/n} \right) \\
&\leq P^n\left( D \in (X \times \mathbb{R})^n : \|f_{D,\lambda} - f_{P,\lambda}\|_H \leq \delta \right).
\end{aligned}
\tag{22}
$$

Let us now assume that we have a training set $D \in (X \times \mathbb{R})^n$ such that $\|f_{P,\lambda} - f_{D,\lambda}\|_H \leq \delta$. Given a pair $(x, y) \in A_{\text{low}}^\delta(f_{P,\lambda})$, we then have
$$\epsilon + \delta < |f_{P,\lambda}(x) - y| \leq |f_{D,\lambda}(x) - y| + |f_{P,\lambda}(x) - f_{D,\lambda}(x)| \leq |f_{D,\lambda}(x) - y| + \delta$$
by the triangle inequality and $\|k\|_\infty \leq 1$ which implies $\|\cdot\|_\infty \leq \|\cdot\|_H$. In other words, we have $A_{\text{low}}^\delta(f_{P,\lambda}) \subset A_{\text{low}}(f_{D,\lambda})$. Consequently, Lemma 3.1 yields
$$
\begin{aligned}
\#SV(f_{D,\lambda}) \geq \#\left\{ i : |f_{D,\lambda}(x_i) - y_i| > \epsilon \right\} &\geq \#\left\{ i : |f_{P,\lambda}(x_i) - y_i| > \epsilon + \delta \right\} \\
&= \sum_{i=1}^n \mathbf{1}_{A_{\text{low}}^\delta(f_{P,\lambda})}(x_i, y_i).
\end{aligned}
$$

Combining this estimate with (22) we then obtain
$$P^n\left( D \in (X \times \mathbb{R})^n : \frac{\#SV(f_{D,\lambda})}{n} \geq \frac{1}{n} \sum_{i=1}^n \mathbf{1}_{A_{\text{low}}^\delta(f_{P,\lambda})}(x_i, y_i) \right) \geq 1 - 3e^{-\frac{\delta^2 \lambda^2 n}{16}}.$$

Moreover, Hoeffding's inequality, see, e.g. [4, Theorem 8.1], shows
$$P^n\left( D \in (X \times \mathbb{R})^n : \frac{1}{n} \sum_{i=1}^n \mathbf{1}_{A_{\text{low}}^\delta(f_{P,\lambda})}(x_i, y_i) > P\left( A_{\text{low}}^\delta(f_{P,\lambda}) \right) - \rho \right) \geq 1 - e^{-2\rho^2 n}.$$

for all $\rho > 0$ and $n \geq 1$. From these estimates and a union bound we conclude the first inequality. In order to show the second estimate we first observe that for training sets $D \in (X \times \mathbb{R})^n$ with $\|f_{\mathrm{P},\lambda} - f_{\mathrm{D},\lambda}\|_H \leq \delta$ we have $A_{\mathrm{up}}(f_{\mathrm{D},\lambda}) \subset A_{\mathrm{up}}^\delta(f_{\mathrm{P},\lambda})$. Lemma 3.1 then shows

$$\#SV(f_{\mathrm{D},\lambda}) \leq \sum_{i=1}^n \mathbf{1}_{A_{\mathrm{up}}^\delta(f_{\mathrm{P},\lambda})}(x_i, y_i),$$

and hence (22) yields

$$\mathrm{P}^n\left(D \in (X \times \mathbb{R})^n : \frac{\#SV(f_{\mathrm{D},\lambda})}{n} \leq \frac{1}{n}\sum_{i=1}^n \mathbf{1}_{A_{\mathrm{up}}^\delta(f_{\mathrm{P},\lambda})}(x_i, y_i)\right) \geq 1 - 3e^{-\frac{\delta^2\lambda^2 n}{16}}.$$

Using Hoeffding's inequality analogously to the proof of the first estimate we then obtain the second estimate. $\blacksquare$

***Proof of of Corollary 2.2:*** We first observe that we have $A_{\mathrm{low}}^\delta(f_{\mathrm{P},\lambda}) \subset A_{\mathrm{low}}^{\delta'}(f_{\mathrm{P},\lambda})$ for $0 \leq \delta' \leq \delta$. Let us show

$$\bigcup_{\delta > 0} A_{\mathrm{low}}^\delta(f_{\mathrm{P},\lambda}) = A_{\mathrm{low}}(f_{\mathrm{P},\lambda}). \tag{23}$$

Obviously, the inclusion "$\subset$" directly follows from the above monotonicity. Conversely, for $(x, y) \in A_{\mathrm{low}}(f_{\mathrm{P},\lambda})$ we have $|f(x) - y| > \epsilon$ and hence $|f(x) - y| > \epsilon + \delta$ for some $\delta > 0$, i.e., we have shown $(x, y) \in A_{\mathrm{low}}^\delta(f_{\mathrm{P},\lambda})$. From (23) we now conclude

$$\lim_{\delta \searrow 0} \mathrm{P}\left(A_{\mathrm{low}}^\delta(f_{\mathrm{P},\lambda})\right) = \mathrm{P}\left(A_{\mathrm{low}}(f_{\mathrm{P},\lambda})\right). \tag{24}$$

In addition, we have $A_{\mathrm{up}}^{\delta'}(f_{\mathrm{P},\lambda}) \subset A_{\mathrm{up}}^\delta(f_{\mathrm{P},\lambda})$ for $0 \leq \delta' \leq \delta$, and it is easy to check that

$$\bigcap_{\delta > 0} A_{\mathrm{up}}^\delta(f_{\mathrm{P},\lambda}) = A_{\mathrm{up}}(f_{\mathrm{P},\lambda}). \tag{25}$$

Indeed, if $(x, y) \in A_{\mathrm{up}}^\delta(f_{\mathrm{P},\lambda})$ for all $\delta > 0$ we have $|f(x) - y| \geq \epsilon - \delta$ for all $\delta > 0$, from which we conclude $|f(x) - y| \geq \epsilon$, i.e. $(x, y) \in A_{\mathrm{up}}(f_{\mathrm{P},\lambda})$. Conversely, the inclusion "$\supset$" directly follows from the above monotonicity of the sets $A_{\mathrm{up}}$. From (25) we then conclude

$$\lim_{\delta \searrow 0} \mathrm{P}\left(A_{\mathrm{up}}^\delta(f_{\mathrm{P},\lambda})\right) = \mathrm{P}\left(A_{\mathrm{up}}(f_{\mathrm{P},\lambda})\right). \tag{26}$$

Let us now fix a decreasing sequence $(\delta_n) \subset (0, 1)$ with $\delta_n \to 0$ and $\delta_n^2 n \to \infty$. Combining (24) and (26) with the estimates of Theorem 2.1, we then obtain the assertion. $\blacksquare$

***Proof of Lemma 2.3:*** Since the loss function $L_\epsilon$ is Lipschitz continuous and convex in $t$, it is easy to verify that $t \mapsto \mathcal{C}_{L_\epsilon,\mathrm{P}(\,\cdot\,|x)}(t)$ is Lipschitz continuous and convex for $\mathrm{P}_X$-almost all $x \in X$, and hence $\mathcal{M}^*(x)$ is a closed interval. In order to prove the remaining assertions it suffices to show that $\lim_{t \to \pm\infty} \mathcal{C}_{L_\epsilon,\mathrm{P}(\,\cdot\,|x)}(t) = \infty$ for $\mathrm{P}_X$-almost all $x \in X$. To this end, we first observe that $\mathcal{R}_{L_\epsilon,\mathrm{P}}^* < \infty$ implies $\mathcal{C}_{L_\epsilon,\mathrm{P}(\,\cdot\,|x)}^* < \infty$ for $\mathrm{P}_X$-almost all $x \in X$. Let us fix such an $x$, a $B > 0$, and a sequence $(t_n) \subset \mathbb{R}$ with $t_n \to -\infty$. By the shape of $L_\epsilon$, there then exists an $r_0 > 0$ such that $L_\epsilon(y, t) \geq 2B$ for all $y, t \in \mathbb{R}$ with $|y - t| \geq r_0$. Furthermore, there exists an $M > 0$ with $\mathrm{P}([-M, M]\,|\,x) \geq 1/2$, and since $t_n \to -\infty$ there further exists an $n_0 \geq 1$ such that $t_n \leq -M - r_0$ for all $n \geq n_0$. For $y \in [-M, M]$ we thus have $y - t_n \geq r_0$, and hence we finally find

$$\mathcal{C}_{L_\epsilon,\mathrm{P}(\,\cdot\,|x)}(t_n) \geq \int_{[-M,M]} L_\epsilon(y, t_n)\, d\mathrm{P}(y|x) \geq B$$

for all $n \geq n_0$. The case $t_n \to \infty$ can be shown analogously. $\blacksquare$

For the proof of Theorem 2.4 we need the following two intermediate results.

**Theorem 3.4** *Let* $\mathrm{P}$ *be a probability measure on* $X \times \mathbb{R}$ *and* $H$ *be a separable RKHS with bounded measurable kernel satisfying* $\|k\|_\infty \leq 1$. *Assume that* $\mathcal{R}_{L_\epsilon,\mathrm{P}}(0) < \infty$ *and that* $H$ *is dense in* $L_1(\mathrm{P}_X)$. *Then, for all* $\delta > 0$ *and* $\rho > 0$, *there exists a* $\lambda_0 > 0$ *such that for all* $\lambda \in (0, \lambda_0]$ *we have*

$$\mathrm{P}_X\left(\{x \in X : |f_{\mathrm{P},\lambda}(x) - t| > \delta \ \text{for all} \ t \in \mathcal{M}^*(x)\}\right) < \rho.$$

***Proof:*** Since $H$ is dense in $L_1(\mathrm{P}_X)$ we have $\inf_{f\in H}\mathcal{R}_{L_\epsilon,\mathrm{P}}(f) = \mathcal{R}^*_{L_\epsilon,\mathrm{P}}$ by [9, Theorem 3], and hence $\lim_{\lambda\to 0}\mathcal{R}_{L_\epsilon,\mathrm{P}}(f_{\mathrm{P},\lambda}) = \mathcal{R}^*_{L_\epsilon,\mathrm{P}}$. Now we obtain the assertion from [6, Theorem 3.16]. ∎

**Lemma 3.5** *Let* $\mathrm{P}$ *be a probability measure on* $X \times \mathbb{R}$ *and* $H$ *be a separable RKHS with bounded measurable kernel satisfying* $\|k\|_\infty \leq 1$. *Assume that* $\mathcal{R}_{L_\epsilon,\mathrm{P}}(0) < \infty$ *and that* $H$ *is dense in* $L_1(\mathrm{P}_X)$. *Then, for all* $\delta > 0$ *and* $\rho > 0$, *there exists a* $\lambda_0 > 0$ *such that for all* $\lambda \in (0,\lambda_0]$ *we have*

$$\mathrm{P}\big(M_{\mathrm{low}}^{2\delta}(f_{\mathrm{P},\lambda})\big) \leq \mathrm{P}\big(A_{\mathrm{low}}^{\delta}(f_{\mathrm{P},\lambda})\big) + \rho \qquad \text{and} \qquad \mathrm{P}\big(M_{\mathrm{up}}^{2\delta}(f_{\mathrm{P},\lambda})\big) \geq \mathrm{P}\big(A_{\mathrm{up}}^{\delta}(f_{\mathrm{P},\lambda})\big) - \rho.$$

***Proof:*** We write $t_\lambda^*(x)$ for the real number defined by (6) for $f(x) := f_{\mathrm{P},\lambda}(x)$. Then we have

$$M_{\mathrm{low}}^{2\delta}(f_{\mathrm{P},\lambda}) \quad \subset \quad \Big(M_{\mathrm{low}}^{2\delta}(f_{\mathrm{P},\lambda}) \cap \big\{(x,y)\in X\times\mathbb{R} : |f_{\mathrm{P},\lambda}(x)-t_\lambda^*(x)|\leq\delta\big\}\Big)$$
$$\cup\big\{(x,y)\in X\times\mathbb{R} : |f_{\mathrm{P},\lambda}(x)-t(x)|>\delta \ \text{ for all } \ t(x)\in\mathcal{M}^*(x)\big\}.$$

Moreover, given an $(x,y)\in M_{\mathrm{low}}^{2\delta}(f_{\mathrm{P},\lambda}) \cap \{(x,y)\in X\times\mathbb{R} : |f_{\mathrm{P},\lambda}(x)-t_\lambda^*(x)|\leq\delta\}$, we find

$$\epsilon + 2\delta < |t_\lambda^*(x)-y| \leq |f_{\mathrm{P},\lambda}(x)-t_\lambda^*(x)| + |f_{\mathrm{P},\lambda}(x)-y| \leq \delta + |f_{\mathrm{P},\lambda}(x)-y|,$$

i.e., we have $(x,y)\in A_{\mathrm{low}}^{\delta}(f_{\mathrm{P},\lambda})$. Estimating the probability of the remaining set by Theorem 3.4 then yields the first assertion. In order to prove the second estimate we first observe that

$$A_{\mathrm{up}}^{\delta}(f_{\mathrm{P},\lambda}) \quad \subset \quad \Big(A_{\mathrm{up}}^{\delta}(f_{\mathrm{P},\lambda}) \cap \big\{(x,y)\in X\times\mathbb{R} : |f_{\mathrm{P},\lambda}(x)-t_\lambda^*(x)|\leq\delta\big\}\Big)$$
$$\cup\big\{(x,y)\in X\times\mathbb{R} : |f_{\mathrm{P},\lambda}(x)-t(x)|>\delta \ \text{ for all } \ t(x)\in\mathcal{M}^*(x)\big\}.$$

For $(x,y)\in A_{\mathrm{up}}^{\delta}(f_{\mathrm{P},\lambda}) \cap \{(x,y)\in X\times\mathbb{R} : |f_{\mathrm{P},\lambda}(x)-t_\lambda^*(x)|\leq\delta\}$ we further have

$$\epsilon - \delta \leq |f_{\mathrm{P},\lambda}(x)-y| \leq |f_{\mathrm{P},\lambda}(x)-t_\lambda^*(x)| + |t_\lambda^*(x)-y| \leq \delta + |t_\lambda^*(x)-y|,$$

i.e., we have $(x,y)\in M_{\mathrm{up}}^{2\delta}(f_{\mathrm{P},\lambda})$. Again, the assertion now follows from Theorem 3.4 . ∎

***Proof of Theorem 2.4:*** Analogously to the proofs of (24) and (26), we find

$$\lim_{\delta\searrow 0}\mathrm{P}\big(M_{\mathrm{low}}^{\delta}(f_{\mathrm{P},\lambda})\big) = \mathrm{P}\big(M_{\mathrm{low}}(f_{\mathrm{P},\lambda})\big) \qquad \text{and} \qquad \lim_{\delta\searrow 0}\mathrm{P}\big(M_{\mathrm{up}}^{\delta}(f_{\mathrm{P},\lambda})\big) = \mathrm{P}\big(M_{\mathrm{up}}(f_{\mathrm{P},\lambda})\big)$$

Combining these equations with Theorem 2.1 and Lemma 3.5, we then obtain the assertion. ∎

## References

[1] A. Christmann and I. Steinwart. Consistency and robustness of kernel based regression. *Bernoulli*, 13:799–819, 2007.

[2] N. Cristianini and J. Shawe-Taylor. *An Introduction to Support Vector Machines*. Cambridge University Press, Cambridge, 2000.

[3] E. De Vito, L. Rosasco, A. Caponnetto, M. Piana, and A. Verri. Some properties of regularized kernel methods. *J. Mach. Learn. Res.*, 5:1363–1390, 2004.

[4] L. Devroye, L. Györfi, and G. Lugosi. *A Probabilistic Theory of Pattern Recognition*. Springer, New York, 1996.

[5] I. Steinwart. Sparseness of support vector machines. *J. Mach. Learn. Res.*, 4:1071–1105, 2003.

[6] I. Steinwart. How to compare different loss functions. *Constr. Approx.*, 26:225–287, 2007.

[7] I. Steinwart and A. Christmann. *Support Vector Machines*. Springer, New York, 2008.

[8] I. Steinwart and A. Christmann. How SVMs can estimate quantiles and the median. In J.C. Platt, D. Koller, Y. Singer, and S. Roweis, editors, *Advances in Neural Information Processing Systems 20*, pages 305–312. MIT Press, Cambridge, MA, 2008.

[9] I. Steinwart, D. Hush, and C. Scovel. Function classes that approximate the Bayes risk. In G. Lugosi and H. U. Simon, editors, *Proceedings of the 19th Annual Conference on Learning Theory*, pages 79–93. Springer, New York, 2006.

[10] V. Vapnik, S. Golowich, and A. Smola. Support vector method for function approximation, regression estimation, and signal processing. In M. Mozer, M. Jordan, and T. Petsche, editors, *Advances in Neural Information Processing Systems 9*, pages 81–287. MIT Press, Cambridge, MA, 1997.

[11] V. N. Vapnik. *Statistical Learning Theory*. John Wiley & Sons, New York, 1998.

[12] V. Yurinsky. *Sums and Gaussian Vectors*. Lecture Notes in Math. 1617. Springer, Berlin, 1995.

[13] T. Zhang. Convergence of large margin separable linear classification. In T. K. Leen, T. G. Dietterich, and V. Tresp, editors, *Advances in Neural Information Processing Systems 13*, pages 357–363. MIT Press, Cambridge, MA, 2001.
